# Can neural networks do better than the Vapnik-Chervonenkis bounds?

**David Cohn**
Dept. of Comp. Sci. & Eng.
University of Washington
Seattle, WA 98195

**Gerald Tesauro**
IBM Watson Research Center
P.O. Box 704
Yorktown Heights, NY 10598

## Abstract

We describe a series of careful numerical experiments which measure the average generalization capability of neural networks trained on a variety of simple functions. These experiments are designed to test whether average generalization performance can surpass the worst-case bounds obtained from formal learning theory using the Vapnik-Chervonenkis dimension (Blumer et al., 1989). We indeed find that, in some cases, the average generalization is significantly better than the VC bound: the approach to perfect performance is exponential in the number of examples $m$, rather than the $1/m$ result of the bound. In other cases, we do find the $1/m$ behavior of the VC bound, and in these cases, the numerical prefactor is closely related to prefactor contained in the bound.

## 1 INTRODUCTION

Probably the most important issue in the study of supervised learning procedures is the issue of generalization, i.e., how well the learning system can perform on inputs not seen during training. Significant progress in the understanding of generalization was made in the last few years using a concept known as the Vapnik-Chervonenkis dimension, or VC-dimension. The VC-dimension provides a basis for a number of powerful theorems which establish worst-case bounds on the ability of arbitrary learning systems to generalize (Blumer et al., 1989; Haussler et al., 1988). These theorems state that under certain broad conditions, the generalization error $\epsilon$ of a learning system with VC-dimension $D$ trained on $m$ random examples of an arbitrary function will with high confidence be no worse than a bound roughly of order $D/m$. The basic requirements for the theorems to hold are that the training

and testing examples are generated from the same probability distribution, and that the learning system is able to correctly classify the training examples.

Unfortunately, since these theorems do not calculate the expected generalization error but instead only bound it, the question is left open whether expected error might lie significantly below the bound. Empirical results of (Ahmad and Tesauro, 1988) indicate that in at least one case, average error was in fact significantly below the VC bound: the error decreased exponentially with the number of examples, $\epsilon \sim \exp(-m/m_0)$, rather than the $1/m$ result of the bound. Also, recent statistical learning theories (Tishby et al., 1989; Schwartz et al., 1990), which provide an analytic means of calculating expected performance, indicate that an exponential approach to perfect performance could be obtained if the spectrum of possible network generalizations has a "gap" near perfect performance.

We have addressed the issue of whether average performance can surpass worst-case performance through numerical experiments which measure the average generalization of simple neural networks trained on a variety of simple functions. Our experiments extend the work of (Ahmad and Tesauro, 1988). They test both the relevance of the worst-case VC bounds to average generalization performance, and the predictions of exponential behavior due to a gap in the generalization spectrum.

## 2   EXPERIMENTAL METHODOLOGY

Two pairs of $N$-dimensional classification tasks were examined in our experiments: two linearly separable functions ("majority" and "real-valued threshold"), and two higher-order functions ("majority-XOR" and "threshold-XOR"). Majority is a Boolean predicate in which the output is 1 if and only if more than half of the inputs are 1. The real-valued threshold function is a natural extension of majority to the continuous space $[0,1]^N$: the output is 1 if and only if the sum of the $N$ real-valued inputs is greater than $N/2$. The majority-XOR function is a Boolean function where the output is 1 if and only if the $N$'th input disagrees with the majority computed by the first $N-1$ inputs. This is a natural extension of majority which retains many of its symmetry properties, e.g., the positive and negative examples are equally numerous and somewhat uniformly distributed. Similarly, threshold-XOR is natural extension of the real-valued threshold function which maps $[0,1]^{N-1} \times \{0,1\} \mapsto \{0,1\}$. Here, the output is 1 if and only if the $N$'th input, which is binary, disagrees with the threshold function computed by the first $N-1$ real-valued inputs. Networks trained on these tasks used sigmoidal units and had standard feed-forward fully-connected structures with at most a single hidden layer. The training algorithm was standard back-propagation with momentum (Rumelhart et al., 1986).

A simulator run consisted of training a randomly initialized network on a training set of $m$ examples of the target function, chosen uniformly from the input space. Networks were trained until all examples were classified within a specified margin of the correct classification. Runs that failed to converge within a cutoff time of 50,000 epochs were discarded. The generalization error of the resulting network was then estimated by testing on a set of 2048 novel examples independently drawn from the same distribution. The average generalization error for a given value of $m$ was typically computed by averaging the results of 10-40 simulator runs, each

with a different set of training patterns, test patterns, and random initial weights. A wide range of values of $m$ was examined in this way in each experiment.

## 2.1    SOURCES OF ERROR

Our experiments were carefully controlled for a number of potential sources of error. Random errors due to the particular choice of random training patterns, test patterns, and initial weights were reduced to low levels by performing a large number of runs and varying each of these in each run.

We have also looked for systematic errors due to the particular values of learning rate and momentum constants, initial random weight scale, frequency of weight changes, training threshold, and training cutoff time. Within wide ranges of parameter values, we find no significant dependence of the generalization performance on the particular choice of any of these parameters except $k$, the frequency of weight changes. (However, the parameter values can affect the rate of convergence or probability of convergence on the training set.) Variations in $k$ appear to alter the numerical coefficients of the learning curve, but not the overall functional form.

Another potential concern is the possibility of overtraining: even though the training set error should decrease monotonically with training time, the test set error might reach a minimum and then increase with further training. We have monitored hundreds of simulations of both the linearly separable and higher-order tasks, and find no significant overtraining in either case.

Other aspects of the experimental protocol which could affect measured results include order of pattern presentation, size of test set, testing threshold, and choice of input representation. We find that presenting the patterns in a random order as opposed to a fixed order improves the probability of convergence, but does not alter the average generalization of runs that do converge. Changing the criterion by which a test pattern is judged correct alters the numerical prefactor of the learning curve but not the functional form. Using test sets of 4096 patterns instead of 2048 patterns has no significant effect on measured generalization values. Finally, convergence is faster with a $[-1, 1]$ coding scheme than with a $[0, 1]$ scheme, and generalization is improved, but only by numerical constants.

## 2.2    ANALYSIS OF DATA

To determine the functional dependence of measured generalization error $\epsilon$ on the number of examples $m$, we apply the standard curve-fitting technique of performing linear regression on the appropriately transformed data. Thus we can look for an exponential law $\epsilon = Ae^{-m/m_0}$ by plotting $\log(\epsilon)$ vs. $m$ and observing whether the transformed data lies on a straight line. We also look for a polynomial law of the form $\epsilon = B/(m + a)$ by plotting $1/\epsilon$ vs. $m$. We have not attempted to fit to a more general polynomial law because this is less reliable, and because theory predicts a $1/m$ law.

By plotting each experimental curve in both forms, $\log(\epsilon)$ vs. $m$ and $1/\epsilon$ vs. $m$, we can determine which model provides a better fit to the data. This can be done both visually and more quantitatively by computing the linear correlation coefficient $r^2$ in a linear least-squares fit. To the extent that one of the curves has a higher value

of $r^2$ than the other one, we can say that it provides a better model of the data than the other functional form.

We have also developed the following technique to assess absolute goodness-of-fit. We generate a set of artificial data points by adding noise equivalent to the error bars on the original data points to the best-fit curve obtained from the linear regression. Regression on the artificial data set yields a value of $r^2$, and repeating this process many times gives a distribution of $r^2$ values which should approximate the distribution expected with the amount of noise in our data. By comparing the value $r^2$ from our original data to this generated distribution, we can estimate the probability that our functional model would produce data like that we observed.

## 3    EXPERIMENTS ON LINEARLY-SEPARABLE FUNCTIONS

Networks with 50 inputs and no hidden units were trained on majority and real-valued threshold functions, with training set sizes ranging from $m = 40$ to $m = 500$ in increments of 20 patterns. Twenty networks were trained for each value of $m$. A total of 3.8% of the binary majority and 7.7% of the real-valued threshold simulation runs failed to converge and were discarded.

The data obtained from the binary majority and real-valued threshold problems was tested for fit to the exponential and polynomial functional models, as shown in Figure 1. The binary majority data had a correlation coefficient of $r^2 = 0.982$ in the exponential fit; this was better than 40% of the "artificial" data sets described previously. However, the polynomial fit only gave a value of $r^2 = 0.966$, which was better than only 6% of the artificial data sets. We conclude that the binary majority data is consistent with an exponential law and not with a $1/m$ law.

The real-valued threshold data, however, behaved in the opposite manner. The exponential fit gave a value of $r^2 = 0.943$, which was better than only 14% of the artificial data sets. However, the polynomial fit gave a value of $r^2 = 0.996$, which was better than 40% of the artificial data sets. We conclude that the real-valued threshold data closely approximates a $1/m$ law and was not likely to have been generated by an exponential law.

## 4    EXPERIMENTS ON HIGHER-ORDER FUNCTIONS

For the majority-XOR and threshold-XOR problems, we used $N = 26$ input units: 25 for the "majority" (or threshold) and a single "XOR" unit. In theory, these problems can be solved with only two hidden units, but in practice, at least three hidden units were needed for reliable convergence. Training set sizes ranging from $m = 40$ to $m = 1000$ in increments of 20 were studied for both tasks. At each value of $m$, 40 simulations were performed. Of the 1960 simulations, 1702 of the binary and 1840 of the real-valued runs converged. No runs in either case achieved a perfect score on the test data.

With both sets of runs, there was a visible change in the shape of the generalization curve when the training set size reached 200 samples. We are interested primarily

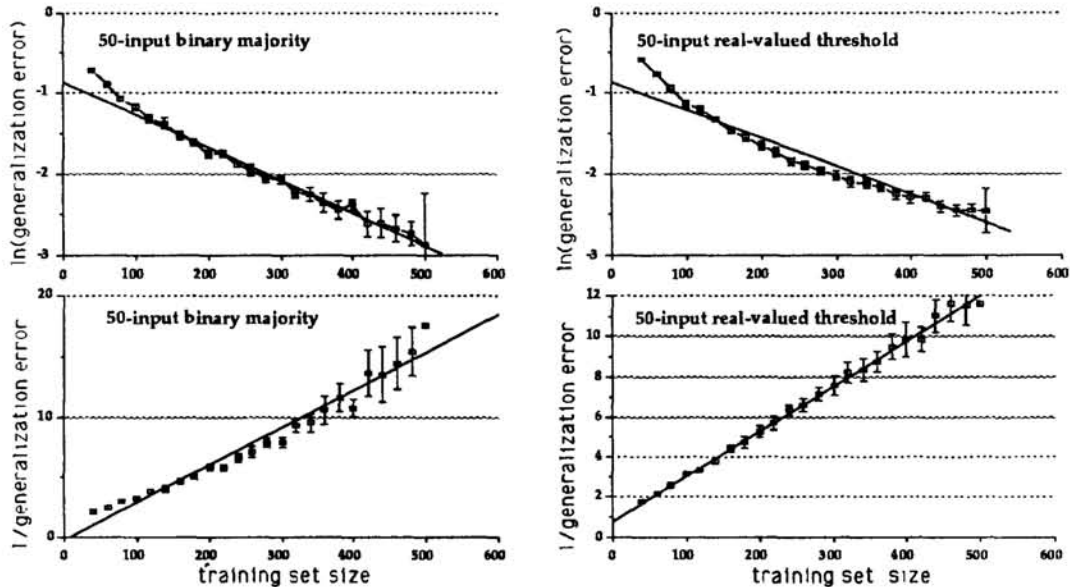

Figure 1: Observed generalization curves for binary majority and real-valued threshold, and their fit to the exponential and polynomial models. Error bars denote 95% confidence intervals for the mean.

in the asymptotic behavior of these curves, so we restricted our analysis to sample sizes 200 and above. As with the single-layer problems, we measured goodness of fit to appropriately linearized forms of the exponential and polynomial curves in question. Results are plotted in Figure 2.

It appears that the generalization curve of the threshold-XOR problem is not likely to have been generated by an exponential, but is a plausible $1/m$ polynomial. The correlation coefficient in the exponential fit is only $r^2 = 0.959$ (better than only 10% of the artificial data sets), but in the polynomial fit is $r^2 = 0.997$ (better than 62% of the artificial data sets).

The binary majority-XOR data, however, appears both visually and from the relative $r^2$ values to fit the exponential model better than the polynomial model. In the exponential fit, $r^2 = 0.994$, while in the polynomial fit, $r^2 = 0.940$. However, we are somewhat cautious because the artificial data test is inconclusive. The exponential fit is better than 40% of artificial data sets, but the polynomial fit is better than 60% of artificial data sets. Also, there appears to be a small component of the curve that is slower than a pure exponential.

## 5 COMPARISON TO THEORY

Figure 3 plots our data for both the first-order and higher-order tasks compared to the thoretical error bounds of (Blumer et al., 1989) and (Haussler et al., 1988). In the higher-order case we have used the total number of weights as an estimate of the VC-dimension, following (Baum and Haussler, 1989). (Even with this low estimate, the bound of (Blumer et al., 1989) lies off the scale.) All of our experimental curves fall below both bounds, and in each case the binary task does asymptotically better than the corresponding real-valued task. One should note that the bound in

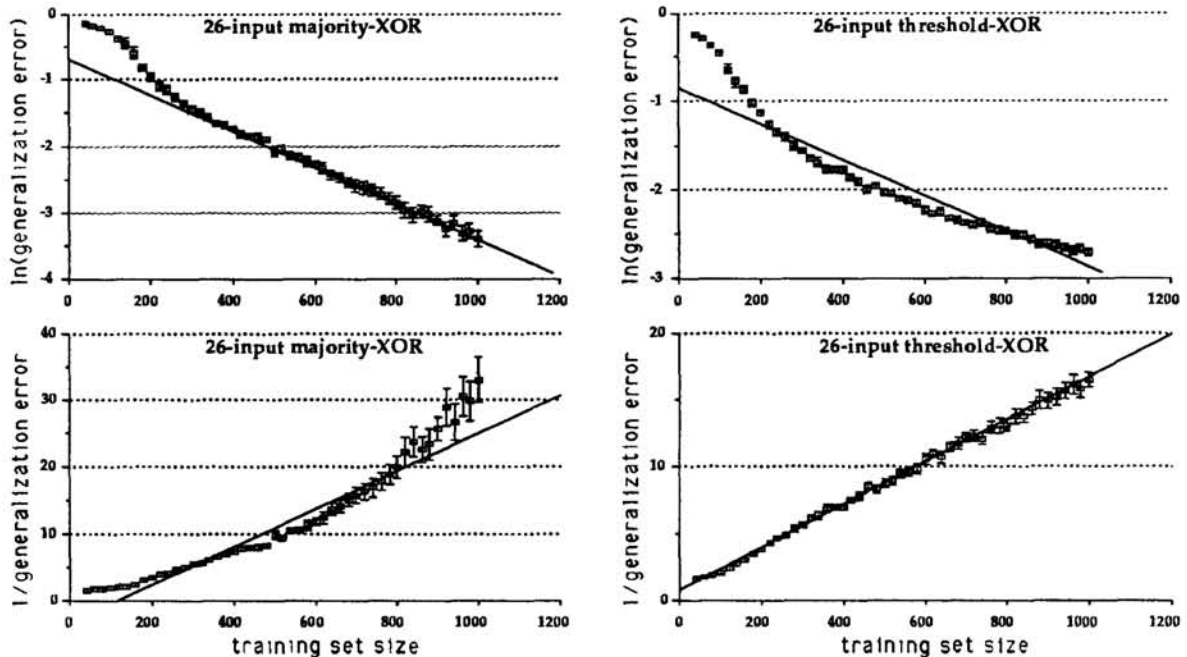

Figure 2: Generalization curves for 26-3-1 nets trained on majority-XOR and threshold-XOR, and their fit to the exponential and polynomial models.

(Haussler et al., 1988) fits the real-valued data to within a small numerical constant. However, strictly speaking it may not apply to our experiments because it is for Bayes-optimal learning algorithms, and we do not know whether back-propagation is Bayes-optimal.

## 6    CONCLUSIONS

We have seen that two problems using strict binary inputs (majority and majority-XOR) exhibited distinctly exponential generalization with increasing training set size. This indicates that there exists a class of problems that is asymptotically much easier to learn than others of the same VC-dimension. This is not only of theoretical interest, but it also has potential bearing on what kinds of large-scale applications might be tractable with network learning methods. On the other hand, merely by making the inputs real instead of binary, we found average error curves lying close to the theoretical bounds. This indicates that the worst-case bounds may be more relevant to expected performance than has been previously realized.

It is interesting that the statistical theories of (Tishby et al., 1989; Schwartz et al., 1990) predict the two classes of behavior seen in our experiments. Our future research will focus on whether or not there is a "gap" as suggested by these theories. Our preliminary findings for majority suggest that there is in fact no gap, except possibly an "inductive gap" in which the learning process for some reason tends to avoid the near-perfect solutions. If such an inductive gap does not exist, then either the theory does not apply to back-propagation, or it must have some other mechanism to generate the exponential behavior.

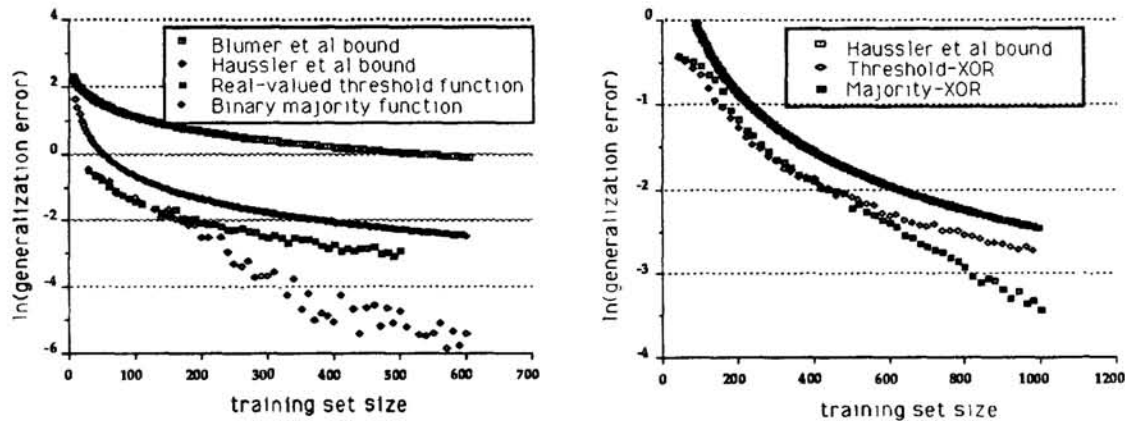

Figure 3: (a) The real-valued threshold problem performs roughly within a constant factor of the upper bounds predicted in (Blumer et al., 1989) and (Haussler et al., 1988), while the binary majority problem performs asymptotically better. (b) The threshold-XOR performs roughly within a constant factor of the predicted bound, while majority-XOR performs asymptotically better.

# References

S. Ahmad and G. Tesauro. (1988) Scaling and generalization in neural networks: a case study. In D. S. Touretzky et al., eds., *Proceedings of the 1988 Connectionist Models Summer School*, 3-10, Morgan Kaufmann.

E. B. Baum and D. Haussler. (1989) What size net gives valid generalization? *Neural Computation* 1(1):151-160.

A. Blumer, A. Ehrenfeucht, D. Haussler, and M. Warmuth. (1989) Learnability and the Vapnik-Chervonenkis dimension. *JACM* 36(4):929-965.

D. Haussler, N. Littlestone, and M. Warmuth. (1990) Predicting {0, 1}-Functions on Randomly Drawn Points. *Tech Report UCSC-CRL-90-54*, Univ. of California at Santa Cruz, CA.

D. E. Rumelhart, G. E. Hinton and R. J. Williams. (1986) Learning internal representations by error propagation. In *Parallel Distributed Processing*, 1:381-362 MIT Press.

D. B. Schwartz, V. K. Samalam, S. A. Solla and J. S. Denker. (1990) Exhaustive learning. *Neural Computation* 2:374-385.

N. Tishby, E. Levin and S. A. Solla. (1989) Consistent inference of probabilities in layered networks: Predictions and generalizations. In *IJCNN Proceedings*, 2:403-409, IEEE.
